# Multitask Learning without Label Correspondences

**Novi Quadrianto**[1], **Alex Smola**[2], **Tibério Caetano**[1], **S.V.N. Vishwanathan**[3], **James Petterson**[1]

1 SML-NICTA & RSISE-ANU, Canberra, ACT, Australia
2 Yahoo! Research, Santa Clara, CA, USA
3 Purdue University, West Lafayette, IN, USA

## Abstract

We propose an algorithm to perform multitask learning where each task has potentially *distinct* label sets and label correspondences are not readily available. This is in contrast with existing methods which either assume that the label sets shared by different tasks are the same or that there exists a label mapping oracle. Our method directly maximizes the mutual information among the labels, and we show that the resulting objective function can be efficiently optimized using existing algorithms. Our proposed approach has a direct application for data integration with different label spaces, such as integrating Yahoo! and DMOZ web directories.

## 1  Introduction

In machine learning it is widely known that if several tasks are related, then learning them simultaneously can improve performance [1–4]. For instance, a personalized spam classifier trained with data from several different users is likely to be more accurate than one that is trained with data from a single user. If one views learning as the task of inferring a function $f$ from the input space $\mathcal{X}$ to the output space $\mathcal{Y}$, then multitask learning is the problem of inferring several functions $f_i : \mathcal{X}_i \mapsto \mathcal{Y}_i$ simultaneously. Traditionally, one either assumes that the set of labels $\mathcal{Y}_i$ for all the tasks are the same (that is, $\mathcal{Y}_i = \mathcal{Y}$ for all $i$), or that we have access to an oracle mapping function $g_{i,j} : \mathcal{Y}_i \mapsto \mathcal{Y}_j$. However, as we argue below, in many natural settings these assumptions are not satisfied.

Our motivating example is the problem of learning to automatically categorize objects on the web into an ontology or directory. It is well established that many web-related objects such as web directories and RSS directories admit a (hierarchical) categorization, and web directories aim to do this in a semi-automated fashion. For instance, it is desirable, when building a categorizer for the Yahoo! directory[1], to take into account other web directories such as DMOZ[2]. Although the tasks are clearly related, their label sets are not identical. For instance, some section heading and sub-headings may be named differently in the two directories. Furthermore, different editors may have made different decisions about the ontology depth and structure, leading to incompatibilities. To make matters worse, these ontologies evolve with time and certain topic labels may die naturally due to lack of interest or expertise while other new topic labels may be added to the directory. Given the large label space, it is unrealistic to expect that a label mapping function is readily available. However, the two tasks are clearly related and learning them simultaneously is likely to improve performance.

This paper presents a method to learn classifiers from a collection of related tasks or data sets, in which each task has its own label dictionary, without constructing an explicit label mapping among them. We formulate the problem as that of maximizing mutual information among the labels sets. We then show that this maximization problem yields an objective function which can be written as a difference of concave functions. By exploiting convex duality [5], we can solve the resulting optimization problem efficiently in the dual space using existing DC programming algorithms [6].

**Related Work** As described earlier, our work is closely related to the research efforts on multitask learning, where the problem of simultaneously learning multiple related tasks is addressed. Several papers have empirically and theoretically highlighted the benefits of multitask learning over single-task learning when the tasks are related. There are several approaches to define task relatedness. The works of [2, 7, 8] consider the setting when the tasks to be learned jointly share a common subset of features. This can be achieved by adding a mixed-norm regularization term that favors a common sparsity profile in features shared by all tasks. Task relatedness can also be modeled as learning functions that are close to each other in some sense [3, 9]. Crammer et al. [10] consider the setting where, in addition to multiple sources of data, estimates of the dissimilarities between these sources are also available. There is also work on data integration via multitask learning where each data source has the same binary label space, whereas the attributes of the inputs can admit different orderings as well as be linearly transformed [11].

The remainder of the paper is organized as follows. We briefly develop background on the maximum entropy estimation problem and its dual in Section 2. We introduce in Section 3 the novel multi-task formulation in terms of a mutual information maximization criterion. Section 4 presents the algorithm to solve the optimization problem posed by the multitask problem. We then present the experimental results, including applications on news articles and web directories data integration, in Section 5. Finally, in Section 6 we conclude the paper.

## 2   Maximum Entropy Duality for Conditional Distributions

Here we briefly summarize the well known duality relation between approximate conditional maximum entropy estimation and maximum a posteriori estimation (MAP) [5, 12]. We will exploit this in Section 4. Recall the definition of the Shannon entropy, $H(y|x) := -\sum_y p(y|x) \log p(y|x)$, where $p(y|x)$ is a conditional distribution on the space of labels $\mathcal{Y}$. Let $x \in \mathcal{X}$ and assume the existence of $\phi(x, y) : \mathcal{X} \times \mathcal{Y} \mapsto \mathcal{H}$, a feature map into a Hilbert space $\mathcal{H}$. Given a data set $(X, Y) := \{(x_1, y_1), \ldots, (x_m, y_m)\}$, where $X := \{x_1, \ldots, x_m\}$, define

$$\mathbf{E}_{y \sim p(y|X)}\left[\phi(X, y)\right] := \frac{1}{m} \sum_{i=1}^m \mathbf{E}_{y \sim p(y|x_i)}\left[\phi(x_i, y)\right], \text{ and } \mu = \frac{1}{m} \sum_{i=1}^m \phi(x_i, y_i). \quad (1)$$

**Lemma 1 ([5], Lemma 6)** *With the above notation we have*

$$\min_{p(y|x)} \sum_{i=1}^m -H(y|x_i) \text{ s.t. } \left\|\mathbf{E}_{y \sim p(y|X)}\left[\phi(X, y)\right] - \mu\right\|_{\mathcal{H}} \le \epsilon \text{ and } \sum_{y \in \mathcal{Y}} p(y|x_i) = 1 \quad (2a)$$

$$= \max_\theta \langle \theta, \mu \rangle_{\mathcal{H}} - \sum_{i=1}^m \log \sum_y \exp(\langle \theta, \phi(x_i, y) \rangle) - \epsilon \|\theta\|_{\mathcal{H}}. \quad (2b)$$

Although we presented a version of the above theorem using Hilbert spaces, it can also be extended to Banach spaces. Choosing different Banach space norms recovers well known algorithms such as $\ell_1$ or $\ell_2$ regularized logistic regression. Also note that by enforcing the moment matching constraint exactly, that is, setting $\epsilon = 0$, we recover the well-known duality between maximum (Shannon) entropy and maximum likelihood (ML) estimation.

## 3   Multitask Learning via Mutual Information

For the purpose of explaining our basic idea, we focus on the case when we want to integrate two data sources such as Yahoo! directory and DMOZ. Associated with each data source are labels $Y = \{y_1, \ldots, y_c\} \subseteq \mathcal{Y}$ and observations $X = \{x_1, \ldots, x_m\} \subseteq \mathcal{X}$ (resp. $Y' = \{y'_1, \ldots, y'_{c'}\} \subseteq \mathcal{Y}'$ and $X' = \{x'_1, \ldots, x'_{m'}\} \subseteq \mathcal{X}'$). The observations are *disjoint* but we assume that they are drawn from the same domain, i.e., $\mathcal{X} = \mathcal{X}'$ (in our running example they are webpages).

If we are interested to solve each of the categorization tasks independently, a maximum entropy estimator described in Section 2 can be readily employed [13]. Here we would like to learn the

two tasks simultaneously in order to improve classification accuracy. Assuming that the labels are different yet correlated we should assume that the joint distribution $p(y, y')$ displays high mutual information between $y$ and $y'$. Recall that the mutual information between random variables $y$ and $y'$ is defined as $I(y, y') = H(y) + H(y') - H(y, y')$, and that this quantity is high when the two variables are mutually dependent. To illustrate this, consider in our running example of integrating Yahoo! and DMOZ web directories, we would expect there is a high mutual dependency between section heading 'Computer & Internet' at Yahoo! directory and 'Computers' at DMOZ directory although they are named somewhat slightly different. Since the marginal distributions over the labels, $p(y)$ and $p(y')$ are fixed, maximizing mutual information can then be viewed as minimizing the joint entropy

$$H(y, y') = -\sum_{y, y'} p(y, y') \log p(y, y'). \tag{3}$$

This reasoning leads us to adding the joint entropy as an additional term for the objective function of the multitask problem. If we define

$$\mu = \frac{1}{m} \sum_{i=1}^{m} \phi(x_i, y_i) \text{ and } \mu' = \frac{1}{m'} \sum_{i=1}^{m'} \phi(x_i', y_i'), \tag{4}$$

then we have the following objective function

$$\underset{p(y|x)}{\text{maximize}} \sum_{i=1}^{m} H(y|x_i) + \sum_{i=1}^{m'} H(y'|x_i') - \lambda H(y, y') \text{ for some } \lambda > 0 \tag{5a}$$

$$\text{s.t. } \left\| \mathbf{E}_{y \sim p(y|X)} \left[ \phi(X, y) \right] - \mu \right\| \le \epsilon \text{ and } \sum_{y \in \mathcal{Y}} p(y|x_i) = 1 \tag{5b}$$

$$\left\| \mathbf{E}_{y' \sim p(y'|X')} \left[ \phi'(X', y') \right] - \mu' \right\| \le \epsilon' \text{ and } \sum_{y' \in \mathcal{Y}'} p(y'|x_i') = 1. \tag{5c}$$

Intuitively, the above objective function tries to find a 'simple' distribution $p$ which is consistent with the observed samples via moment matching constraints while also taking into account task relatedness. We can recover the single task maximum entropy estimator by removing the joint entropy term (by setting $\lambda = 0$), since the optimization problem (the objective functions as well as the constraints) in (5) will be decoupled in terms of $p(y|x)$ and $p(y'|x')$. There are two main challenges in solving (5):

- The joint entropy term $H(y, y')$ is concave, hence the above objective of the optimization problem is not concave in general (it is the difference of two concave functions). We therefore propose to solve this non-concave problem using DC programming [6], in particular the concave convex procedure (CCCP) [14, 15].
- The joint distribution between labels $p(y, y')$ is unknown. We will estimate this quantity (therefore the joint entropy quantity) from the observations $x$ and $x'$. Further, we assume that $y$ and $y'$ are conditionally independent given an arbitrary input $x \in \mathcal{X}$, that is $p(y, y'|x) = p(y|x)p(y'|x)$. For instance, in our example, annotations made by an editor at Yahoo! and an editor at DMOZ on the set of webpages are assumed conditionally independent given the set of webpages. This assumption essentially means that the labeling process depends entirely on the set of webpages, i.e., any other latent factors that might connect the two editors are ignored.

In the following section we discuss in further detail how to address these two challenges, as well as the resulting optimization problem obtained, which can be solved efficiently by existing convex solvers.

## 4 Optimization

The concave convex procedure (CCCP) works as follow: for a given function $f(x) = g(x) - h(x)$, where $g$ is concave and $-h$ is convex, a lower bound can be found by

$$f(x) \ge g(x) - h(x_0) - \langle \partial h(x_0), x - x_0 \rangle. \tag{6}$$

This lower bound is concave and can be maximized effectively over a convex domain. Subsequently one finds a new location $x_0$ and the entire procedure is repeated. This procedure is guaranteed to converge to a local optimum or saddle point [16].

Therefore, one potential approach to solve the optimization problem in (5) is to use successive linear lower bounds on $H(y, y')$ and to solve the resulting decoupled problems in $p(y|x)$ and $p(y'|x')$ separately. We estimate the joint entropy term $H(y, y')$ by its empirical quantity on $x$ and $x'$ with the conditional independence assumption (in the sequel, we make the dependency of $p(y|x)$ on a parameter $\theta$ explicit and similarly for the dependency of $p(y'|x')$ on $\theta'$), that is

$$H(y, y'|X) = -\sum_{y,y'} \left[ \frac{1}{m} \sum_{i=1}^{m} p(y|x_i, \theta) p(y'|x_i, \theta') \right] \log \left[ \frac{1}{m} \sum_{j=1}^{m} p(y|x_j, \theta) p(y'|x_j, \theta') \right], \quad (7)$$

and similarly for $H(y, y'|X')$. Each iteration of CCCP approximates the convex part (negative joint entropy) by its tangent, that is $\langle \partial h(x_0), x \rangle$ in (6). Therefore, taking derivatives of the joint entropy with respect to $p(y|x_i)$ and evaluating at parameters at iteration $t-1$, denoted as $\theta_{t-1}$ and $\theta'_{t-1}$, yields

$$g_y(x_i) := -\partial_{p(y|x_i)} H(y, y'|X) \tag{8}$$

$$= \frac{1}{m} \sum_{y'} \left[ 1 + \log \frac{1}{m} \sum_{j=1}^{m} p(y|x_j, \theta_{t-1}) p(y'|x_j, \theta'_{t-1}) \right] p(y'|x_i, \theta'_{t-1}). \tag{9}$$

Define similarly $g_y(x'_i)$, $g_{y'}(x_i)$, and $g_{y'}(x'_i)$ for the derivative with respect to $p(y|x'_i)$, $p(y'|x_i)$ and $p(y'|x'_i)$, respectively. This leads, by optimizing the lower bound in (6), to the following decoupled optimization problems in $p(y|x_i)$ and an analogous problem in $p(y'|x'_i)$:

$$\min_{p(y|x)} \sum_{i=1}^{m} \left[ -H(y|x_i) + \lambda \sum_y g_y(x_i) p(y|x_i) \right] + \sum_{i=1}^{m'} \left[ -H(y|x'_i) + \lambda' \sum_y g_y(x'_i) p(y|x'_i) \right] \tag{10a}$$

$$\text{subject to} \quad \left\| \mathbf{E}_{y \sim p(y|X)}[\phi(X, y)] - \mu \right\| \leq \epsilon. \tag{10b}$$

The above objective function is still in the form of maximum entropy estimation, with the linearization of the joint entropy quantities acting like additional evidence terms. Furthermore, we also impose an additional maximum entropy requirement on the 'off-set' observations $p(y|x'_i)$, as after all we also want the 'simplicity' requirement of the distribution $p$ on the input $x'_i$. We can of course weigh the requirement on 'off-set' observations differently.

While we succeed in reducing the non-concave objective function in (5) to a decoupled concave objective function in (10), it might be desirable to solve the problem in the dual space due to difficulty in handling the constraint in (10b). The following lemma shows the duality of the objective function in (10). The proof is given in the supplementary material.

**Lemma 2** *The corresponding Fenchel's dual of (10) is*

$$\min_\theta \sum_{i=1}^{m} \log \sum_y \exp(\langle \theta, \phi(x_i, y) \rangle - \lambda g_y(x_i)) + \sum_{i=1}^{m'} \log \sum_y \exp(\langle \theta, \phi(x'_i, y) \rangle - \lambda' g_y(x'_i))$$

$$- \frac{1}{m} \sum_{i=1}^{m} \langle \theta, \phi(x_i, y_i) \rangle + \epsilon \|\theta\|_{\ell_2} \tag{11}$$

The above dual problem still has the form of logistic regression with the additional evidence terms from task relatedness appearing in the log-partition function. Several existing convex solvers can be used to solve the optimization problem in (11) efficiently. Refer to Algorithm 1 for a pseudocode of our proposed method.

**Initialization** For each iteration of CCCP, the linearization part of the joint entropy function requires the value of $\theta$ and $\theta'$ at the previous iteration (refer to (9)). At the beginning of the iteration, we can start the algorithm with a uniform prior, i.e. set $p(y) = 1/|\mathcal{Y}|$ and $p(y') = 1/|\mathcal{Y}'|$.

---

**Algorithm 1** Multitask Mutual Information

---

**Input:** Datasets $(X, Y)$ and $(X', Y')$ with $\mathcal{Y} \neq \mathcal{Y}'$, number of iterations $N$
**Output:** $\theta, \theta'$
Initialize $p(y) = 1/|\mathcal{Y}|$ and $p(y') = 1/|\mathcal{Y}'|$
**for** $t = 1$ to $N$ **do**
   Solve the dual problem in (11) w.r.t. $p(y|x, \theta)$ and obtain $\theta_t$
   Solve the dual problem in (11) w.r.t. $p(y'|x', \theta')$ and obtain $\theta'_t$
**end for**
**return** $\theta \leftarrow \theta_N, \theta' \leftarrow \theta'_N$

---

## 5 Experiments

To assess the performance of our proposed multitask algorithm, we perform binary n-task ($n \in \{3, 5, 7, 10\}$) experiments on MNIST digit dataset and a multiclass 2-task experiment on the Reuters1-v2 dataset plus an application on integrating Yahoo! and DMOZ web directory. We detail those experiments in turn in the following sections.

### 5.1 MNIST

**Datasets** MNIST data set[3] consists of $28 \times 28$-size images of hand-written digits from 0 through 9. We use a small sample of the available training set to simulate the situation when we only have limited number of labeled examples and test the performance on the entire available test set. In this experiment, we look at a binary n-task ($n \in \{3, 5, 7, 10\}$) problem. We consider digits $\{8, 9, 0\}$, $\{6, 7, 8, 9, 0\}$, $\{4, 5, 6, 7, 8, 9, 0\}$ and $\{1, 2, 3, 4, 5, 6, 7, 8, 9, 0\}$ for the 3-task, 5-task, 7-task and 10-task, respectively. To simulate the problem that we have distinct label dictionaries for each task, we consider the following setting: in the 3-task problem, the first task has binary labels $\{+1, -1\}$, where label $+1$ means digit 8 and label $-1$ means digit 9 and 0; in the second task, label $+1$ means digit 9 and label $-1$ means digit 8 and 0; lastly in the third task, label $+1$ means digit 0 and label $-1$ means digit 8 and 9. Similar one-against-rest grouping is also used for 5-task, 7-task and 10-task problems. Each of the tasks has its *own* input $x$.

**Algorithms** We couldn't find in the literature of multitask learning methods addressing the same problem as the one we study: learn multiple tasks when there is no correspondence between the output spaces. Therefore we compared the performance of our multitask method against the baseline given by the maximum entropy estimator applied to each of the tasks independently. Note that we focus on the setting in which data sources have disjoint sets of covariate observations (vide Section 3) and thus a simple strategy of multilabel prediction with union of label sets corresponds to our baseline. For both ours and the baseline method, we use a Gaussian kernel to define the implicit feature map on the inputs. The width of the kernel was set to the median between pairs of observations, as suggested in [17]. The regularization parameter was tuned for the single task estimator and the same value was used for the multitask. The weight on the joint entropy term was set to be equal to 1.

**Pairwise Label Correlation** Section 3 describes the multitask objective function for the case of the 2-task problem. For the case when the number of tasks to be learned jointly is greater than 2, we experiment in two different ways: in one approach we can define the joint entropy term on the full joint distribution, that is when we want to learn jointly 3 different tasks having label $y$, $y'$ and $y''$, we can then define the joint entropy as $H(y, y', y'') = -\sum_{y, y', y''} p(y, y', y'') \log p(y, y', y'')$. As more computationally efficient way, we can consider the joint entropy on the pairwise distribution instead. We found that the performance of our method is quite similar for the two cases and we report results only on the pairwise case.

**Results** The experiments are repeated for 10 times and the results are summarized in Table 1. We find that, on average, jointly learning the multiple related tasks *always* improves the classification

Table 1: Performance assessment, Accuracy $\pm$ STD. $m(m')$ denotes the number of training data points (number of test points). `STL`: single task learning; `MTL`: multi task learning and `Upper Bound`: multi class learning. **Boldface** indicates a significance difference between STL and MTL (one-sided paired Welch t-test with 99.95% confidence level).

| Tasks | m (m') | STL | MTL | Upper Bound |
|---|---|---|---|---|
| 8 \-8 | 15 (2963) | 77.39±5.23 | **80.03±4.83** | 93.42±0.87 |
| 9 \-9 | 15 (2963) | 91.12±5.94 | 91.96±5.42 | 95.99±0.75 |
| 0 \-0 | 120 (2963) | 98.66±0.67 | 98.21±0.92 | 98.79±0.25 |
| Average | | 89.06 | 90.07 | 96.07 |
| | | | | |
| 6 \-6 | 25 (4949) | 81.79±10.18 | **83.86±9.51** | 96.37±1.06 |
| 7 \-7 | 25 (4949) | 70.73±16.58 | **72.84±15.77** | 91.99±2.23 |
| 8 \-8 | 25 (4949) | 62.52±10.15 | **66.77±9.43** | 92.05±1.76 |
| 9 \-9 | 25 (4949) | 63.80±13.70 | **67.26±12.65** | 92.53±1.65 |
| 0 \-0 | 150 (4949) | **97.35±1.33** | 96.60±1.64 | 97.59±0.62 |
| Average | | 75.84 | 77.47 | 94.10 |
| | | | | |
| 4 \-4 | 70 (6823) | 71.69±6.83 | **73.49±6.77** | 91.20±1.55 |
| 5 \-5 | 70 (6823) | 67.55±4.70 | **70.10±4.61** | 89.30±0.34 |
| 6 \-6 | 70 (6823) | 86.31±2.93 | **87.21±2.77** | 94.03±0.95 |
| 7 \-7 | 70 (6823) | 83.34±3.54 | 84.02±3.69 | 91.94±0.90 |
| 8 \-8 | 70 (6823) | 75.61±6.00 | 76.97±5.12 | 87.46±1.69 |
| 9 \-9 | 70 (6823) | 63.69±11.42 | 65.74±10.15 | 86.89±1.79 |
| 0 \-0 | 210 (6823) | **97.20±1.49** | 96.56±1.67 | 97.24±0.73 |
| Average | | 77.91 | 79.16 | 91.15 |
| | | | | |
| 1 \-1 | 100 (10000) | 96.59±2.11 | 96.80±1.91 | 96.89±0.59 |
| 2 \-2 | 100 (10000) | 67.77±3.49 | **69.95±2.68** | 88.74±1.94 |
| 3 \-3 | 100 (10000) | 72.59±5.90 | **74.18±5.54** | 87.59±2.95 |
| 4 \-4 | 100 (10000) | 69.91±5.82 | 71.76±5.47 | 92.87±0.94 |
| 5 \-5 | 100 (10000) | 53.78±2.78 | **57.26±2.72** | 85.71±1.38 |
| 6 \-6 | 100 (10000) | 79.22±5.21 | 80.54±4.53 | 92.93±0.98 |
| 7 \-7 | 100 (10000) | 76.57±10.2 | 77.18±9.43 | 89.83±1.24 |
| 8 \-8 | 100 (10000) | 63.57±2.65 | **65.85±2.50** | 83.51±0.63 |
| 9 \-9 | 100 (10000) | 63.28±6.69 | **65.38±6.09** | 84.94±1.45 |
| 0 \-0 | 300 (10000) | **98.43±0.84** | 97.81±1.01 | 98.49±0.40 |
| Average | | 74.17 | 75.67 | 90.82 |

accuracy. When assessing the performance on each of the tasks, we notice that the advantage of learning jointly is particularly significant for those tasks with smaller number of observations.

## 5.2 Ontology

**News Ontologies** In this experiment, we consider multiclass learning in a 2-task problem. We use the Reuters1-v2 news article dataset [18] which has been pre-processed[4]. In the pre-processing stage, the label hierarchy is reorganized by mapping the data set to the second level of topic hierarchy. The documents that only have labels of the third or fourth levels are mapped to their parent category of the second level. The documents that only have labels of the first level are not mapped onto any category. Lastly any multi-labelled instances are removed. The second level hierarchy consists of 53 categories and we perform experiments on the top 10 categories. TF-IDF features are used, and the dictionary size (feature dimension) is 47236. For this experiment, we use 12500 news articles to form one set of data and another 12500 news article to form the second set of data. In the first set, we group the news articles having the label $\{1, 2\}$, $\{3, 4\}$, $\{5, 6\}$, $\{7, 8\}$ and $\{9, 10\}$ and re-label it as $\{1, 2, 3, 4, 5\}$. For the second set of data, it also has 5 labels but this time the labels are

Table 2: Yahoo! Top Level Categorization Results. `STL:` single task learning accuracy; `MTL:` multi task learning accuracy; `% Imp.:` relative performance improvement. The highest relative improvement at Yahoo! is for the topic of *'Computer & Internet'*, i.e. there is an increase in accuracy from 48.12% to 52.57%. Interestingly, DMOZ has a similar topic but was called *'Computers'* and it achieves accuracy of 75.72%.

| Topic | MTL/STL | (% Imp.) | Topic | MTL/STL | (% Imp.) |
|---|---|---|---|---|---|
| Arts | 56.27/55.11 | (2.10) | News & Media | 15.23/14.83 | (1.03) |
| Business & Economy | 66.52/66.88 | (-0.53) | Recreation | 68.81/67.00 | (2.70) |
| Computer & Internet | 52.57/48.12 | (9.25) | Reference | 26.65/24.81 | (7.42) |
| Education | 62.48/63.02 | (-0.85) | Regional | 62.85/61.86 | (1.60) |
| Entertainment | 63.30/61.37 | (3.14) | Science | 78.58/79.75 | (-1.46) |
| Government | 24.44/22.88 | (6.82) | Social Science | 31.55/30.68 | (2.84) |
| Health | 85.42/85.27 | (1.76) | Society & Culture | 49.51/49.05 | (0.94) |

Table 3: DMOZ Top Level Categorization Results. `STL:` single task learning accuracy; `MTL:` multi task learning accuracy; `% Imp.:` relative performance improvement. The improvement of multitask to single task on each topic is negligible for DMOZ web directories. Arguably, this can be partly explained as DMOZ has higher average topic categorization accuracy than Yahoo! and there might be more knowledge to be shared from DMOZ to Yahoo! than vice versa.

| Topic | MTL/STL | (% Imp.) | Topic | MTL/STL | (% Imp.) |
|---|---|---|---|---|---|
| Arts | 57.52/57.84 | (-0.5) | Reference | 67.42/67.42 | (0) |
| Business | 54.02/53.05 | (1.83) | Regional | 28.59/28.56 | (0.10) |
| Computers | 75.08/75.72 | (-0.8) | Science | 42.67/42.09 | (1.38) |
| Games | 78.58/78.58 | (0) | Shopping | 75.20/74.62 | (0.54) |
| Health | 82.34/82.55 | (-0.14) | Society | 57.68/58.20 | (-0.89) |
| Home | 67.47/67.47 | (0) | Sports | 83.49/83.53 | (-0.05) |
| News | 61.70/62.01 | (-0.49) | World | 87.80/87.57 | (0.26) |
| Recreation | 58.04/58.25 | (-0.36) | | | |

generated by $\{1, 6\}$, $\{2, 7\}$, $\{3, 8\}$, $\{4, 9\}$ and $\{5, 10\}$ grouping. We split equally the news articles on each set to form training and test sets. We run a maximum entropy estimator independently, $p(y|x, \theta)$ and $p(y'|x', \theta')$ , on the two sets achieving accuracy of $92.59\%$ for the first set and $91.53\%$ for the second set. We then learn the two sets of the news articles jointly and in the first test set, we achieve accuracy of $93.81\%$. For the second test set, we achieve an accuracy of $93.31\%$. This experiment further emphasizes that it is possible to learn several related tasks simultaneously even though they have different label sets and it is beneficial to do so.

**Web Ontologies** We also perform an experiment on the data integration of Yahoo! and DMOZ web directories. We consider the top level of the Yahoo!'s topic tree and sample web links listed in the directory. Similarly we also consider the top level of the DMOZ topic tree and retrieve sampled web links. We consider the content of the first page of each web link as our input data. It is possible that the first page that is being linked from the web directory contain mostly images (for the purpose of attracting visitors), thus we only consider those webpages that have enough texts to be a valid input. This gives us $19186$ webpages for Yahoo! and $35270$ for DMOZ. For the sake of getting enough texts associated with each link, we can actually crawl many more pages associated with the link. However, we find that it is quite damaging to do so because as we crawl deeper the topic of the texts are rapidly changing. We use the standard bag-of-words representation with TF-IDF weighting as our features. The dictionary size (feature dimension) is $27075$. We then use 2000 web pages from Yahoo! and 2000 pages from DMOZ as training sets and the remainder as test sets. Table 2 and 3 summarize the experimental results.

From the experimental results on web directories integration, we observe the following:

- Similarly to the experiments on MNIST digits and Reuters1-v2 news articles, multitask learning always helps on *average*, i.e. the average relative improvements are positive for both Yahoo! and DMOZ web directories;

- The improvement of multitask to single task on each topic is more prominent for Yahoo! web directories and is negligible for DMOZ web directories (2.62% and 0.07%, respectively). Arguably, this can be partly explained as Yahoo! has lower average topic categorization accuracy than DMOZ (c.f. 60.22% and 64.68 %, respectively). It seems that there is much more knowledge to be shared from DMOZ to Yahoo! in the hope to increase the latter's classification accuracies;

- Looking closely at accuracy at each topic, the highest relative improvement at Yahoo! is for the topic of *'Computer & Internet'*, i.e. there is an increase in accuracy from 48.12% to 52.57%. Interestingly, DMOZ has a similar topic but was called *'Computers'* and it achieves accuracy of 75.72%. The improvement might be partly because our proposed method is able to discover the implicit label correlations despite the two topics being named differently;

- Regarding the worst classified categories, we have *'News & Media'* for Yahoo! and *'Regional'* for DMOZ. This is intuitive since those two topics can indeed cover a wide range of subjects. The easiest category to be classified is *'Health'* for Yahoo! and *'World'* for DMOZ. As well, this is quite intuitive as the world of health contains mostly specific jargon and the world of world has much language-specific webpage content.

## 6 Discussion and Conclusion

We presented a method to learn classifiers from a collection of related tasks or data sets, in which each task has its own label set. Our method works without the need of an explicit mapping between the label spaces of the different tasks. We formulate the problem as one of maximizing the mutual information among the label sets. Our experiments on binary $n$-task ($n \in \{3, 5, 7, 10\}$) and multiclass 2-task problems revealed that, on average, jointly learning the multiple related tasks, albeit with different label sets, always improves the classification accuracy. We also provided experiments on a prototypical application of our method: classifying in Yahoo! and DMOZ web directories. Here we deliberately used small amounts of data–a common situation in commercial tagging and classification. This shows that classification accuracy of Yahoo! significantly increased. Given that DMOZ classification was already 4.5% better prior to the application of our method, this shows the method was able to transfer classification accuracy from the DMOZ task to the Yahoo! task. Furthermore, the experiments seem to suggest that our proposed method is able to discover implicit label correlations despite the lack of label correspondences.

Although the experiments on web directories integration is encouraging, we have clearly only touched the surface of possibilities to be explored. While we focused on the categorization at the top level of the topic tree, it might be beneficial (and further highlight the usefulness of multitask learning, as observed in [2–4, 9]) to consider categorization at deeper levels (take for example the second level of the tree), where we have much fewer observations for each category. In the extreme case, we might consider the labels as corresponding to a directed acyclic graph (DAG) and encode the feature map associated with the label hierarchy accordingly. One instance as considered in [19] is to use a feature map $\phi(y) \in \mathbb{R}^k$ for $k$ nodes in the DAG (excluding the root node) and associate with every label $y$ the vector describing the path from the root node to $y$, ignoring the root node itself.

Furthermore, the application of data integration which admit a hierarchical categorization goes beyond web related objects. With our method, it is also now possible to learn classifiers from a collection of related gene-ontology graphs [20] or patent hierarchies [19].

**Acknowledgments**    NICTA is funded by the Australian Government as represented by the Department of Broadband, Communications and the Digital Economy and the Australian Research Council through the ICT Centre of Excellence program. N. Quadrianto is partly supported by Microsoft Research Asia Fellowship.

## Footnotes

[1] http://dir.yahoo.com/

[2] http://www.dmoz.org/

[3] http://yann.lecun.com/exdb/mnist

[4]http://www.csie.ntu.edu.tw/~cjlin/libsvmtools/datasets/multiclass.html

# References

[1] R. Caruana. Multitask learning. *Machine Learning*, 28:41–75, 1997.

[2] Andreas Argyriou, Theodoros Evgeniou, and Massimiliano Pontil. Convex multi-task feature learning. *Mach. Learn.*, 73(3):243–272, 2008.

[3] Kai Yu, Volker Tresp, and Anton Schwaighofer. Learning gaussian processes from multiple tasks. In *ICML '05: Proceedings of the 22nd international conference on Machine learning*, pages 1012–1019, New York, NY, USA, 2005. ACM.

[4] Rie Kubota Ando and Tong Zhang. A framework for learning predictive structures from multiple tasks and unlabeled data. *Journal of Machine Learning Research*, 6:1817–1853, 2005.

[5] Y. Altun and A.J. Smola. Unifying divergence minimization and statistical inference via convex duality. In H.U. Simon and G. Lugosi, editors, *Proc. Annual Conf. Computational Learning Theory*, LNCS, pages 139–153. Springer, 2006.

[6] T. Pham Dinh and L. Hoai An. A D.C. optimization algorithm for solving the trust-region subproblem. *SIAM Journal on Optimization*, 8(2):476–505, 1988.

[7] G. Obozinski, B. Taskar, and M. I. Jordan. Multi-task feature selection. Technical report, U.C. Berkeley, 2007.

[8] Remi Flamary, Alain Rakotomamonjy, Gilles Gasso, and Stephane Canu. Svm multi-task learning and non convex sparsity measure. In *The Learning Workshop*, 2009.

[9] Theodoros Evgeniou, Charles A. Micchelli, and Massimiliano Pontil. Learning multiple tasks with kernel methods. *J. Mach. Learn. Res.*, 6:615–637, 2005.

[10] K. Crammer, M. Kearns, and J. Wortman. Learning from multiple sources. In *NIPS 19*, pages 321–328. MIT Press, 2007.

[11] Shai Ben-David, Johannes Gehrke, and Reba Schuller. A theoretical framework for learning from a pool of disparate data sources. In *KDD '02: Proceedings of the 8th ACM international conference on Knowledge discovery and data mining*, pages 443–449. ACM, 2002.

[12] M. Dudík and R. E. Schapire. Maximum entropy distribution estimation with generalized regularization. In Gábor Lugosi and Hans U. Simon, editors, *Proc. Annual Conf. Computational Learning Theory*. Springer Verlag, June 2006.

[13] Nadia Ghamrawi and Andrew McCallum. Collective multi-label classification. In *CIKM '05: Proceedings of the 14th ACM international conference on Information and knowledge management*, pages 195–200, New York, NY, USA, 2005. ACM.

[14] A.L. Yuille and A. Rangarajan. The concave-convex procedure. *Neural Computation*, 15:915–936, 2003.

[15] A. J. Smola, S. V. N. Vishwanathan, and T. Hofmann. Kernel methods for missing variables. In R.G. Cowell and Z. Ghahramani, editors, *Proceedings of International Workshop on Artificial Intelligence and Statistics*, pages 325–332, 2005.

[16] Bharath Sriperumbudur and Gert Lanckriet. On the convergence of the concave-convex procedure. In Y. Bengio, D. Schuurmans, J. Lafferty, C. K. I. Williams, and A. Culotta, editors, *Advances in Neural Information Processing Systems 22*, pages 1759–1767. MIT Press, 2009.

[17] B. Schölkopf. *Support Vector Learning*. R. Oldenbourg Verlag, Munich, 1997. Download: http://www.kernel-machines.org.

[18] David D. Lewis, Yiming Yang, Tony G. Rose, and Fan Li. RCV1: A new benchmark collection for text categorization research. *The Journal of Machine Learning Research*, 5:361–397, 2004.

[19] Lijuan Cai and T. Hofmann. Hierarchical document categorization with support vector machines. In *Proceedings of the Thirteenth ACM conference on Information and knowledge management*, pages 78–87, New York, NY, USA, 2004. ACM Press.

[20] M. Ashburner, C. A. Ball, J. A. Blake, D. Botstein, H. Butler, J. M. Cherry, A. P. Davis, K. Dolinski, S. S. Dwight, J. T. Eppig, M. A. Harris, D. P. Hill, L. Issel-Tarver, A. Kasarskis, S. Lewis, J. C. Matese, J. E. Richardson, M. Ringwald, G. M. Rubin, and G. Sherlock. Gene ontology: tool for the unification of biology. the gene ontology consortium. *Nat Genet*, 25:25–29, 2000.

[21] J. M. Borwein and Q. J. Zhu. *Techniques of Variational Analysis*. CMS books in Mathematics. Canadian Mathematical Society, 2005.

